# An Efficient Implementation of the Back-propagation Algorithm on the Connection Machine CM-2

Xiru Zhang[1]   Michael Mckenna   Jill P. Mesirov   David L. Waltz
*Thinking Machines Corporation*
*245 First Street, Cambridge, MA 02142-1214*

## ABSTRACT

In this paper, we present a novel implementation of the widely used Back-propagation neural net learning algorithm on the Connection Machine CM-2 – a general purpose, massively parallel computer with a hypercube topology. This implementation runs at about 180 million interconnections per second (IPS) on a 64K processor CM-2. The main interprocessor communication operation used is 2D nearest neighbor communication. The techniques developed here can be easily extended to implement other algorithms for layered neural nets on the CM-2, or on other massively parallel computers which have 2D or higher degree connections among their processors.

## 1   Introduction

High-speed simulation of large artificial neural nets has become an important tool for solving real world problems and for studying the dynamic behavior of large populations of interconnected processing elements [3, 2]. This work is intended to provide such a simulation tool for a widely used neural net learning algorithm – the Back-propagation (BP) algorithm.[7]

The hardware we have used is the Connection Machine® CM-2.[2] On a 64K processor CM-2 our implementation runs at 40 million *Weight Update Per Second*

(WUPS)[3] for training, or 180 million *Interconnection Per Second* (IPS) for forward-pass, where IPS is defined in the DARPA NEURAL NETWORK STUDY [2] as *"the number of multiply-and-add operations that can be performed in a second" [on a Back-propagation network]*. We believe that the techniques developed here can be easily extended to implement other algorithms for layered neural nets on the CM-2, or other massively parallel machines which have 2D or higher degree connections among their processors.

## 2     The Connection Machine

The Connection Machine CM-2 is a massively parallel computer with up to $65,536$ processors. Each processor has a single-bit processing unit and 64K or 256K bits of local RAM. The processors run in SIMD mode. They are connected in an $n$-cube topology, which permits highly efficient $n$ dimensional grid communications. The system software also provides *scan* and *spread* operations – e.g., when $n \cdot m$ processors are connected as an $n \times m$ 2D grid, the summation (product, max, *etc.*) of a "parallel variable" value in all the processors on a row of the grid[4] takes only $O(\log m)$ time. It is possible to turn off any subset of the processors so that instructions will only be performed by those processors that are currently active. On the CM-2, every 32 processors share a floating point processing unit; and a 32 bit number can be stored across 32 processors (i.e., one bit per processor). These 32 processors can each access this 32-bit number as if it were stored in its own memory. This is a way of sharing data among processors locally. The CM-2 uses a conventional computer such as a SUN-4, VAX or Symbolics Lisp Machine as a front-end machine. Parallel extensions to the familiar programming languages LISP, C, and FORTRAN, via the front-end, allow the user to program the Connection Machine and the front-end system.

## 3     The Back-propagation Algorithm

The Back-propagation [7] algorithm works on layered, feed-forward networks (*BP net* for short in the following discussion), where the processing units are arranged in layers – there are an input layer, an output layer, and one or more "hidden layers" (layers between the input and output layers). A BP net computes its output in the following fashion: first an input pattern is set as the output of the units at the input layer; then one layer at a time, from the input to hidden to output layer, the units compute their outputs by applying an activation function to the weighted sum of their inputs (which are the outputs of the unit at the lower layer(s) that are connected to them). The weights come from the links between the units.

The Back-propagation algorithm "trains" a BP net by adjusting the link weights of the net using a set of "training examples." Each training example consists of

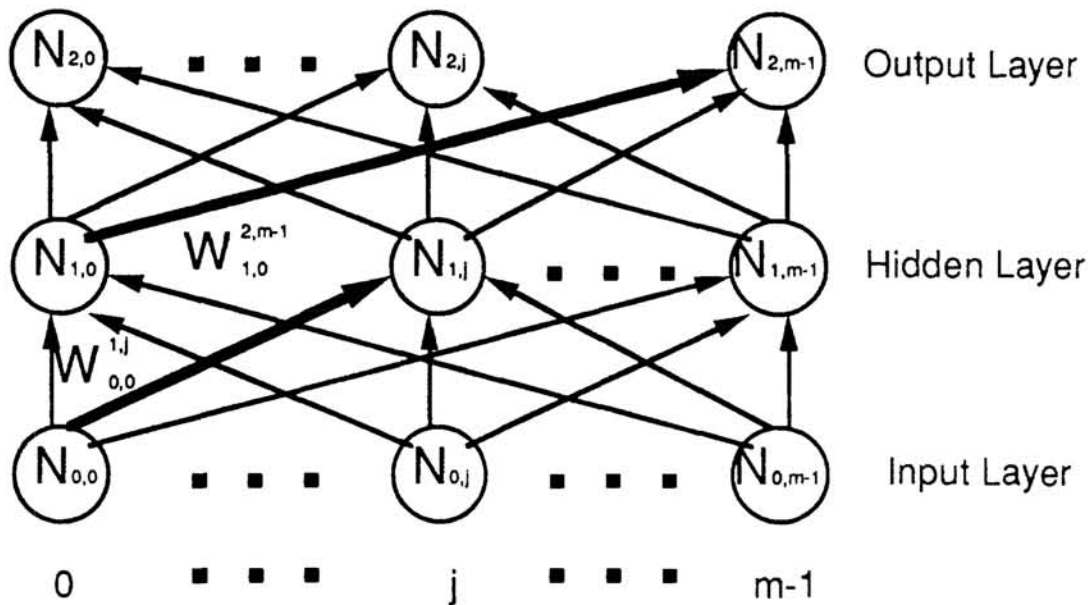

**Figure 1:** A 3-layer, fully-connected Back-propagation network that has the same number ($m$) of nodes at each layer.

an input pattern and an ideal output pattern that the user wants the network to produce for that input. The weights are adjusted based on the difference between the ideal output and the actual output of the net. This can be seen as a gradient descent process in the weight space.

After the training is done, the BP net can be applied to inputs that are not in the set of training examples. For a new input pattern IP, the network tends to produce an output similar to the training example whose input is similar to IP. This can be used for interpolation, approximation, or generalization from examples depending on the goal of the user [4].

## 4   The Implementation

In this section, we explain our implementation by presenting a simple example – a three-layer fully-connected BP network that has the same number of nodes at each layer. It is straightforward to extend it to general cases. For a more detailed discussion, see reference [8].

### 4.1   A Simple Case

Figure 1 shows a fully-connected 3-layer BP network with $m$ nodes on each layer. In the following discussion, we will use $N_{i,j}$ to denote the $j$th node (from the left) on layer $i$, $i \in \{0, 1, 2\}$, $j \in \{0, 1, \ldots, m-1\}$; $W_{k,h}^{i,j}$ is the weight of the link from node $N_{k,h}$ to node $N_{i,j}$, and $\delta_{i,j}$ is the error at node $N_{i,j}$.

First, assume we have exactly $m$ processors. We store a "column" of the network in each processor. That is, processor $j$ contains nodes $N_{0,j}$, $N_{1,j}$ and $N_{2,j}$. It also contains the weights of the links going into $N_{1,j}$ and $N_{2,j}$ (i.e., $W_{0,k}^{1,j}$ and $W_{1,k}^{2,j}$ for

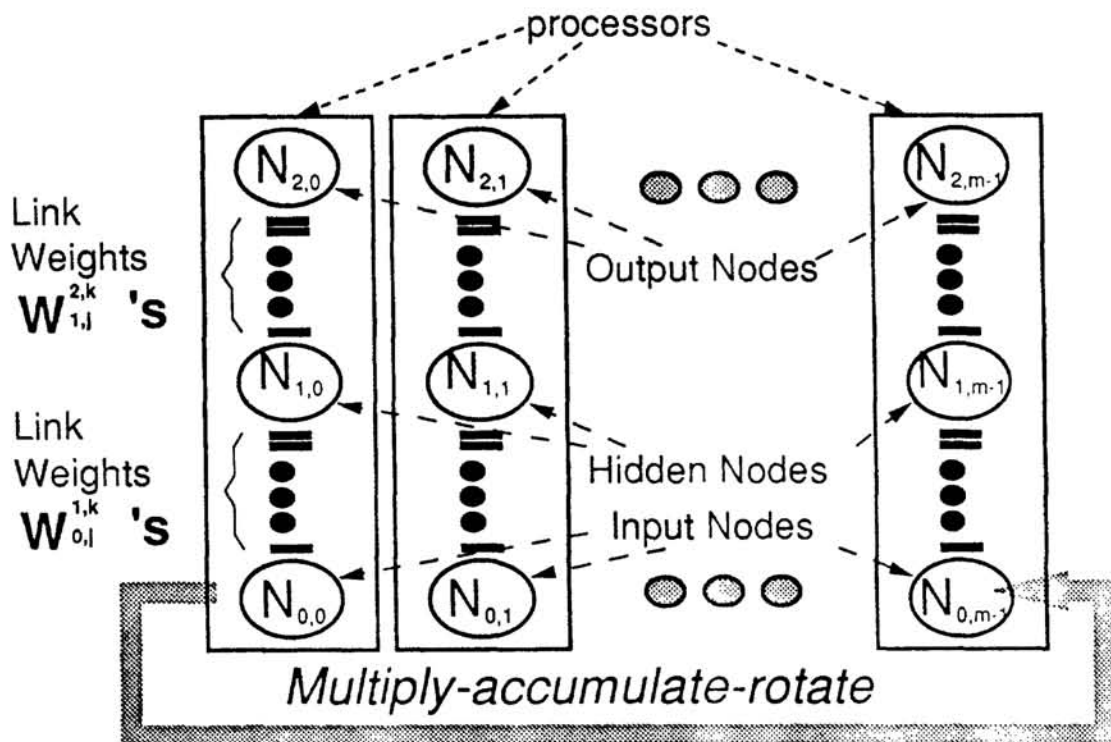

**Figure 2:** The layout of the example network.

$k \in \{0, 1, \ldots, m-1\}$). See **Figure 2**. The Back-propagation algorithm consists of three steps: (1) *forward pass* to compute the network output; (2) *backward propagation* to compute the errors at each node; and (3) *weight update* to adjust the weights based on the errors. These steps are implemented as follows:

**4.1.1**    **Forward Pass:** $Output(N_{i,j}) = F(\sum_{k=0}^{m-1} W_{i-1,k}^{ij} \cdot Output(N_{i-1,k}))$

We implement *forward pass* as follows:

1. Set the input node values; there is one input node per processor.

2. In each processor, multiply the input node value by the link weight between the input node and the hidden node that is in the same processor; then accumulate the product in the hidden node.

3. Rotate the input node values – each processor sends its input node value to its nearest left neighbor processor, the leftmost processor sends its value to the rightmost processor; i.e., do a left-circular-shift.

4. Repeat the *multiply-accumulate-rotate* cycles in the above two steps (2–3) $m$ times; every hidden node $N_{1,j}$ will then contain $\sum_{k=0}^{m-1} W_{0,k}^{1j} \cdot Output(N_{0,k})$. Now apply the activation function $F$ to that sum. (See **Figure 2**.)

5. Repeat steps 2–4 for the output layer, using the hidden layer as the input.

### 4.1.2   Backward Propagation

For the output layer, $\delta_{2,k}$, the error at each node $N_{2,k}$, is computed by

$$\delta_{2,k} = Output(N_{2,k}) \cdot (1 - Output(N_{2,k})) \cdot (Target(N_{2,k}) - Output(N_{2,k})),$$

where $Target(N_{2,k})$ is the ideal output for node $N_{2,k}$. This error can be computed in place, i.e., no inter-processor communication is needed. For the hidden layer,

$$\delta_{1,j} = Output(N_{1,j}) \cdot (1 - Output(N_{1,j})) \cdot \sum_{k=0}^{m-1} W_{1,j}^{2,k} \cdot \delta_{2,k}$$

To compute $\sum_{k=0}^{m-1} W_{1,j}^{2,k} \cdot \delta_{2,k}$ for the hidden nodes, we perform a *multiply-accumulate-rotate* operation similar to the *forward pass*, but from the top down. Notice that the weights between a hidden node and the output nodes are in different processors. So, instead of rotating $\delta_{2,k}$'s at the output layer, we rotate the partial sum of products for the hidden nodes: at the beginning every hidden node $N_{1,j}$ has an accumulator $A_j$ with initial value $= 0$ in processor $j$. We do a left-circular-shift on the $A_j$'s. When $A_j$ moves to processor $k$, we set $A_j \leftarrow A_j + W_{1,j}^{2,k} \cdot \delta_{2,k}$. After $m$ rotations, $A_j$ will return to processor $j$ and its value will be $\sum_{k=0}^{m-1} W_{1,j}^{2,k} \cdot \delta_{2,k}$.

### 4.1.3   Weight Update:  $\Delta W_{k,h}^{i,j} = \eta \cdot \delta_{i,j} \cdot Output(N_{k,h})$

$\Delta W_{k,h}^{i,j}$ is the weight increment for $W_{k,h}^{i,j}$, $\eta$ is the "learning rate" and $\delta_{i,j}$ is the error for node $N_{i,j}$, which is computed in the *backward propagation* step and is stored in processor $j$. The *weight update* step is done as follows:

1. In each processor $j$, for the weights between the input layer and hidden layer, compute weight update $\Delta W_{0,k}^{1,j} = \eta \cdot \delta_{1,j} \cdot Output(N_{0,k})$,[5] and add $\Delta W_{0,k}^{1,j}$ to $W_{0,k}^{1,j}$;[6]

2. Rotate the input node values as in step 3 of the *forward pass*.

3. Repeat the above two steps $m$ times, until all the weights between the input layer and the hidden layer are updated.

4. Do the above for weights between the hidden layer and the output layer also.

We can see that the basic operation is the same for all three steps of the Back-propagation algorithm, i.e., *multiply-accumulate-rotate*. On the CM-2, *multiply*, *add* (for accumulate) and *circular-shift* (for rotate) take roughly the same amount of time, independent of the size of the machine. So the CM-2 spends only about $1/3$ of its total time doing communication in our implementation.

## 4.2    Replication of Networks

Usually, there are more processors on the CM-2 than the width of a BP network. Suppose the network width is $m$ and there are $n \cdot m$ processors; then we make $n$ copies of the network on the CM-2, and do the *forward pass* and *backward propagation* for *different* training patterns on each copy of the network. For the *weight update* step, we can sum up the weight changes from different copies of the network (i.e. from different training patterns), then update the weights in all the copies by this sum. This is equivalent to updating the weights after $n$ training patterns on a single copy of the BP network.

On the CM-2, every 32 processors can share the same set of data (see section 2). We make use of this feature and store the BP network weights across sets of 32 processors. Thus each processor only needs to allocate one bit for each weight. Also, since the weight changes from different training patterns are additive, there is no need to add them up in advance – each copy of the network can update (add to) the weights separately, as long as no two or more copies of the network update the same weight at the same time. (Our implementation guarantees that no such weight update conflict can occur.) See Figure 3.

We call the 32 copies of the network that share the same set of weights a *block*. When the number of copies $n > 32$, say $n = 32 \cdot q$, then there will be $q$ blocks on the CM-2. We need to sum up the weight changes from different blocks before updating the weights in each block. This summation takes a very small portion of the total running time (much less than 1%). So the time increase can usually be ignored when there is more than one block.[7] Thus, the implementation speeds up essentially linearly as the number of processors increases.

## 5    An Example: Character Image Recovery

In this example, a character, such as A, is encoded as a $16 \times 16$ pixel array. A 3-layer fully-connected network with 256 input nodes, 128 hidden nodes and 256 output nodes is trained with 64 character pixel arrays, each of which is used both as the input pattern and the ideal output pattern. After the training is done ($maximum\_error < 0.15$),[8] some noisy character images are fed into the network. The network is then used to remove the noise (to recover the images). We can also use the network recursively – to feed the network output back as the input.

Figure 4a shows the ideal outputs (odd columns) and the actual outputs (even columns) of the network after the training. Figure 4b shows corrupted character image inputs (odd columns) and the recovered images (even columns). The corrupted inputs have 30% noise, i.e., 30% of the pixels take random values in each image. We can see that most of the characters are recovered.

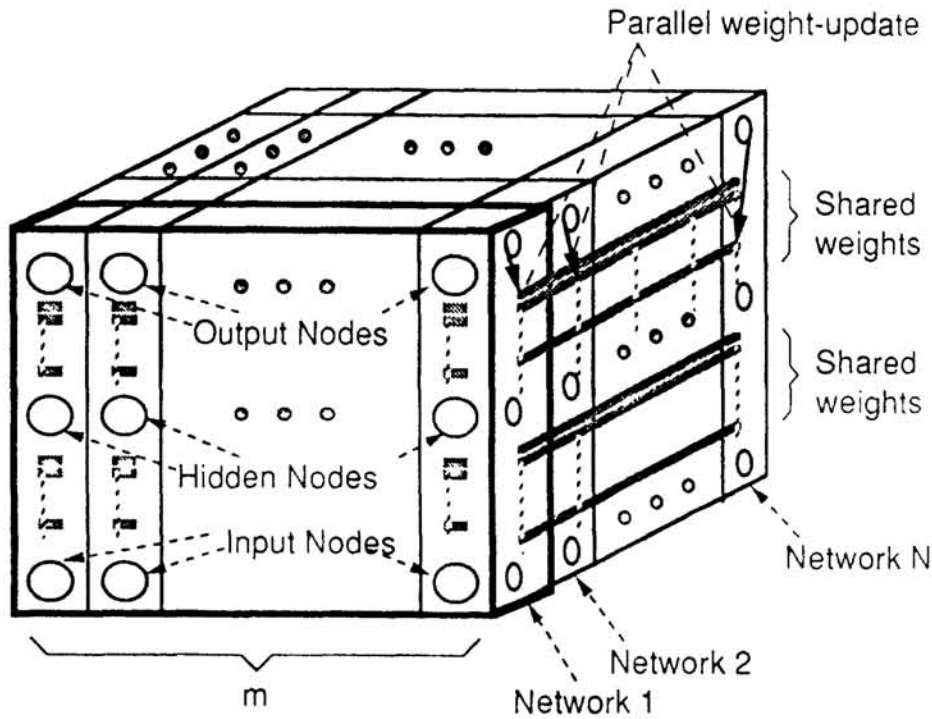

**Figure 3:** Replication of a BP network and parallel update of network weights. In the *weight update* step, the nodes in each copy of the BP network loop through the weights going into them in the following fashion: in the first loop, Network 1 updates the first weight, Network 2 updates the second weight ... Network $N$ updates the $N$th weight; in general, in the $J$th loop, Network I updates $[Mod(I + J, N)]th$ weight. In this way, it is guaranteed that no two networks update the same weight at the same time. When the total number of weights going into each node is greater than $N$, we repeat the above loop.

```
AAaaBBbbTTttUUuu        AAaaBBbbTTttUUuu
CCcoDDddVVvvXXxx        CCccDDddVVvvXXxx
EEeeFFffYYyyZZzz        EEesFFffYYyyZZzz
GG99HHhhOO1122 33      GG99HHhhOO1122 33
IIiiKKkk44556677        IIiAKKkk44556677
LLllNNnn8899<<>>        LLlINNnn8899<<>>
OOooPPPP??$$^^&&        OOooPPPP??$$^^&&
RRrrSSss**++==--        RRrrSSss**++==%Y
```

<div align="center">(a)                              (b)</div>

**Figure 4:** (a) Ideal outputs (in odd columns) and the actual after-training outputs (in even columns) of a network with 256 input nodes, 128 hidden nodes and 256 output nodes trained with character images. (b) Noisy inputs (in odd columns) and the corresponding outputs ("cleaned-up" images) produced by the network.

| Computer | BP performance (IPS) |
| --- | --- |
| CM-2 | 180 M |
| Cray X-MP | 50 M |
| WARP (10) | 17 M (WUPS) |
| ANZA plus | 10 M |
| TRW MK V (16) | 10 M |
| Butterfly (64) | 8 M |
| SAIC SIGMA-1 | 5-8 M |
| TI Odyessy | 5 M |
| Convex C-1 | 3.6 M |
| VAX 8600 | 2 M |
| SUN 3 | 250 K |
| Symbolics 3600 | 35 K |

**Table 1:** Comparison of BP implementations on different computers.

In this example, we used a 4K processor CM-2. The BP network had $256 \times 128 + 128 \times 256 = 65,536$ weights. We made 64 copies of the network on the CM-2, so there were 2 blocks. One weight update cycle[9] took 1.66 seconds. Thus the performance is: $(65,536 \times 64) \div 1.66 \approx 2,526,689$ weight update per second (WUPS). Within the 1.66 seconds, the communication between the two blocks took 0.0023 seconds. If we run a network of the same size on a 64K processor CM-2,[10] there will be 32 blocks, and the inter-block communication will take $0.0023 \times \frac{\log 32}{\log 2} = 0.0115$ second.[11] And the overall performance will be:

$(16 \times 65,536 \times 64) \div (1.66 + 0.0115) = 40,148,888$ WUPS

*Forward-pass* took 22% of the total time. Thus if we ran the forward pass alone, the speed would be $40,148,888 \div 0.22 \approx 182,494,940$ IPS.

## 6    Comparison With Other Implementations

This implementation of the Back-propagation algorithm on the CM-2 runs much more efficiently than previous CM implementations (e.g., see [1], [6]). Table 1 lists the speeds of Back-propagation on different machines (obtained from reference [2] and [5]).

# 7   Summary

In this paper, we have shown an example of efficient implementation of neural net algorithms on the Connection Machine CM-2. We used Back-propagation because it is the most widely implemented, and many researchers have used it as a benchmark. The techniques developed here can be easily adapted to implement other algorithms on layered neural nets.

The main communication operation used in this work is the 2D grid nearest neighbor communication. The facility for a group of processors on the CM-2 to share data is important in reducing the amount of space required to store network weights and the communication between different copies of the network. These points should be kept in mind when one tries to use the techniques described here on other machines.

The main lesson we learned from this work is that to implement an algorithm efficiently on a massively parallel machine often requires re-thinking of the algorithm to explore the parallel nature of the algorithm, rather than just a straightforward translation of serial implementations.

### Acknowledgement

Many thanks to Alex Singer, who read several drafts of this paper and helped improve it. Lennart Johnsson helped us solve a critical problem. Discussions with other members of the Mathematical and Computational Sciences Group at Thinking Machines Corporation also helped in many ways.

## Footnotes

[1] This author is also a graduate student at Computer Science Department, Brandeis University, Waltham, MA 02254-9110.

[2]  Connection Machine is a registered trademark of Thinking Machines Corporation.

[3]  This includes the time required to read in the input pattern, propagate activation forward through the network, read in the ideal output pattern, propagate the error signal backward through the network, compute the weight changes, and change the weights.

[4]  That is, to add together one value from each processor on a row of the grid and distribute the sum into all the processors on the same row.

[5]   Initially $k = j$, but the input node values will be rotated around in later steps, so $k \neq j$ in general.

[6]   $W_{0,k}^{1,j}$ is in the same processor as $\Delta W_{0,k}^{1,j}$ – all the weights going into node $N_{1,j}$ are in processor $j$. Also we can accumulate $\Delta W_{0,k}^{1,j}$ for several training patterns instead of updating $W_{0,k}^{1,j}$ every time. We can also keep the previous weight change and add a "momentum" term here. (Our implementation actually does all these. They are omitted here to simplify the explanation of the basic ideas.)

[7] The summation is done using the *scan* and *spread* operations (see section 2), so its time increases only logarithmically in proportion to the number of blocks. Usually there are only a few blocks, thus we could use the nearest neighbor communication here instead without much loss of performance.

[8] This training took about 400 cycles.

[9] See footnote 3 for definition.

[10] Assume we have enough training patterns to fill up the CM-2.

[11] We use *scan* and *spread* operations here, so the time used increases logrithmatically.

# References

[1] Louis G. Ceci, Patrick Lynn, and Phillip E. Gardner. Efficient Distribution of Back-Propagation Models on Parallel Architectures. Tech. Report CU-CS-409-88, Dept. of Computer Science, University of Colorado, September 1988.

[2] MIT Lincoln Laboratory. Darpa Neural Network Study. Final Report, July 1988.

[3] Special Issue on Artificial Neural Systems. IEEE Computer, March 1988.

[4] Tomaso Poggio and Federico Girosi. A Theory of Networks for Approximation and Learning. A.I.Memo 1140, MIT AI Lab, July 1989.

[5] Dean A. Pomerleau, George L. Gusciora David S. Touretzky, and H. T. Kung. Neural Network Simulation at Warp Speed: How We Got 17 Million Connections Per Second. In *IEEE Int. Conf. on Neural Networks*, July 1988. San Diego, CA.

[6] Charles R. Rosenberg and Guy Blelloch. An Implementation of Network Learning on the Connection Machine. In *Proceedings of the Tenth International Joint Conference on Artificial Intelligence*, Milan, Italy, 1987.

[7] D. E. Rumelhart, G. E. Hinton, and R. J. Williams. Learning internal representations by error propagation. In *Parallel Distributed Processing*, chapter 8. MIT Press, 1986.

[8] Xiru Zhang, Michael Mckenna, Jill P. Mesirov, and David L. Waltz. An Efficient Implementation of The Back-Propagation Algorithm On the Connection Machine CM-2. Technical Report RL-89-1, Thinking Machines Corp., 245 First St. Cambridge, MA 02114, 1989.